# CMOL CrossNets: Possible Neuromorphic Nanoelectronic Circuits

**Jung Hoon Lee**          **Xiaolong Ma**          **Konstantin K. Likharev**
Stony Brook University
Stony Brook, NY 11794-3800
*klikharev@notes.cc.sunysb.edu*

## Abstract

Hybrid "CMOL" integrated circuits, combining CMOS subsystem with nanowire crossbars and simple two-terminal nanodevices, promise to extend the exponential Moore-Law development of microelectronics into the sub-10-nm range. We are developing neuromorphic network ("CrossNet") architectures for this future technology, in which neural cell bodies are implemented in CMOS, nanowires are used as axons and dendrites, while nanodevices (bistable latching switches) are used as elementary synapses. We have shown how CrossNets may be trained to perform pattern recovery and classification despite the limitations imposed by the CMOL hardware. Preliminary estimates have shown that CMOL CrossNets may be extremely dense (~$10^7$ cells per cm$^2$) and operate approximately a million times faster than biological neural networks, at manageable power consumption. In Conclusion, we discuss in brief possible short-term and long-term applications of the emerging technology.

## 1   Introduction: CMOL Circuits

Recent results [1, 2] indicate that the current VLSI paradigm based on CMOS technology can be hardly extended beyond the 10-nm frontier: in this range the sensitivity of parameters (most importantly, the gate voltage threshold) of silicon field-effect transistors to inevitable fabrication spreads grows exponentially. This sensitivity will probably send the fabrication facilities costs skyrocketing, and may lead to the end of Moore's Law some time during the next decade.

There is a growing consensus that the impending Moore's Law crisis may be preempted by a radical paradigm shift from the purely CMOS technology to hybrid CMOS/nanodevice circuits, e.g., those of "CMOL" variety (Fig. 1). Such circuits (see, e.g., Ref. 3 for their recent review) would combine a level of advanced CMOS devices fabricated by the lithographic patterning, and two-layer nanowire crossbar formed, e.g., by nanoimprint, with nanowires connected by simple, similar, two-terminal nanodevices at each crosspoint. For such devices, molecular single-electron latching switches [4] are presently the leading candidates, in particular because they may be fabricated using the self-assembled monolayer (SAM) technique which already gave reproducible results for simpler molecular devices [5].

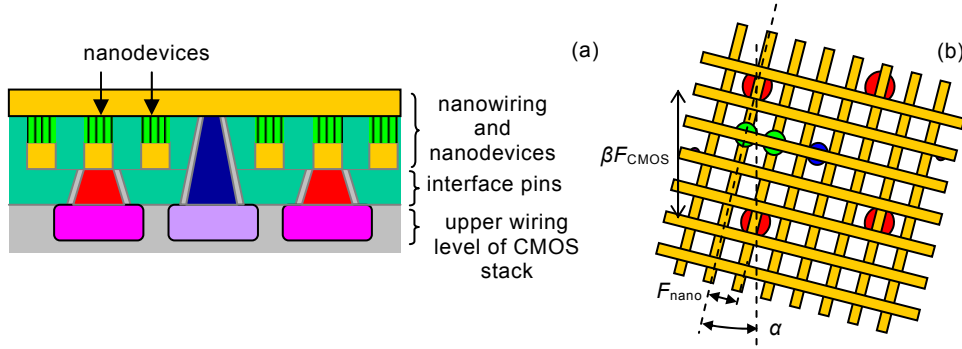

Fig. 1. CMOL circuit: (a) schematic side view, and (b) top-view zoom-in on several adjacent interface pins. (For clarity, only two adjacent nanodevices are shown.)

In order to overcome the CMOS/nanodevice interface problems pertinent to earlier proposals of hybrid circuits [6], in CMOL the interface is provided by pins that are distributed all over the circuit area, on the top of the CMOS stack. This allows to use advanced techniques of nanowire patterning (like nanoimprint) which do not have nanoscale accuracy of layer alignment [3]. The vital feature of this interface is the tilt, by angle $\alpha = \arcsin(F_{nano}/\beta F_{CMOS})$, of the nanowire crossbar relative to the square arrays of interface pins (Fig. 1b). Here $F_{nano}$ is the nanowiring half-pitch, $F_{CMOS}$ is the half-pitch of the CMOS subsystem, and $\beta$ is a dimensionless factor larger than 1 that depends on the CMOS cell complexity. Figure 1b shows that this tilt allows the CMOS subsystem to address each nanodevice even if $F_{nano} \ll \beta F_{CMOS}$.

By now, it has been shown that CMOL circuits can combine high performance with high defect tolerance (which is necessary for any circuit using nanodevices) for several digital applications. In particular, CMOL circuits with defect rates below a few percent would enable terabit-scale memories [7], while the performance of FPGA-like CMOL circuits may be several hundred times above that of overcome purely CMOL FPGA (implemented with the same $F_{CMOS}$), at acceptable power dissipation and defect tolerance above 20% [8].

In addition, the very structure of CMOL circuits makes them uniquely suitable for the implementation of more complex, mixed-signal information processing systems, including ultradense and ultrafast neuromorphic networks. The objective of this paper is to describe in brief the current status of our work on the development of so-called Distributed Crossbar Networks ("CrossNets") that could provide high performance despite the limitations imposed by CMOL hardware. A more detailed description of our earlier results may be found in Ref. 9.

## 2   Synapses

The central device of CrossNet is a two-terminal latching switch [3, 4] (Fig. 2a) which is a combination of two single-electron devices, a transistor and a trap [3]. The device may be naturally implemented as a single organic molecule (Fig. 2b). Qualitatively, the device operates as follows: if voltage $V = V_j - V_k$ applied between the external electrodes (in CMOL, nanowires) is low, the trap island has no net electric charge, and the single-electron transistor is closed. If voltage $V$ approaches certain threshold value $V_+ > 0$, an additional electron is inserted into the trap island, and its field lifts the Coulomb blockade of the single-electron transistor, thus connecting the nanowires. The switch state may be reset (e.g., wires disconnected) by applying a lower voltage $V < V_- < V_+$.

Due to the random character of single-electron tunneling [2], the quantitative description of the switch is by necessity probabilistic: actually, $V$ determines only the rates $\Gamma_{\uparrow\downarrow}$ of device

switching between its ON and OFF states. The rates, in turn, determine the dynamics of probability $p$ to have the transistor opened (i.e. wires connected):

$$dp/dt = \Gamma_\uparrow(1 - p) - \Gamma_\downarrow p. \tag{1}$$

The theory of single-electron tunneling [2] shows that, in a good approximation, the rates may be presented as

$$\Gamma_{\uparrow\downarrow} = \Gamma_0 \exp\{\pm e(V - S)/k_B T\} , \tag{2}$$

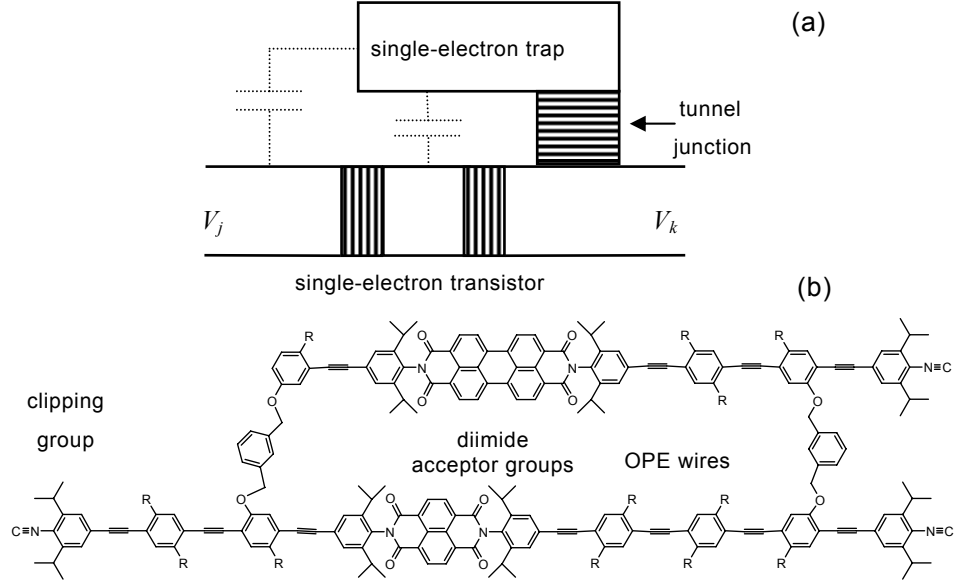

Fig. 2. (a) Schematics and (b) possible molecular implementation of the two-terminal single-electron latching switch

where $\Gamma_0$ and $S$ are constants depending on physical parameters of the latching switches. Note that despite the random character of switching, the strong nonlinearity of Eq. (2) allows to limit the degree of the device "fuzziness".

## 3 CrossNets

Figure 3a shows the generic structure of a CrossNet. CMOS-implemented somatic cells (within the Fire Rate model, just nonlinear differential amplifiers, see Fig. 3b,c) apply their output voltages to "axonic" nanowires. If the latching switch, working as an elementary synapse, on the crosspoint of an axonic wire with the perpendicular "dendritic" wire is open, some current flows into the latter wire, charging it. Since such currents are injected into each dendritic wire through several (many) open synapses, their addition provides a natural passive analog summation of signals from the corresponding somas, typical for all neural networks. Examining Fig. 3a, please note the open-circuit terminations of axonic and dendritic lines at the borders of the somatic cells; due to these terminations the somas do not communicate directly (but only via synapses).

The network shown on Fig. 3 is evidently feedforward; recurrent networks are achieved in the evident way by doubling the number of synapses and nanowires per somatic cell (Fig. 3c). Moreover, using dual-rail (bipolar) representation of the signal, and hence doubling the number of nanowires and elementary synapses once again, one gets a CrossNet with

somas coupled by compact 4-switch groups [9]. Using Eqs. (1) and (2), it is straightforward to show that that the average synaptic weight $w_{jk}$ of the group obeys the "quasi-Hebbian" rule:

$$\frac{d}{dt}\left\langle w_{jk} \right\rangle = -4\Gamma_0 \sinh\left(\gamma S\right)\sinh\left(\gamma V_j\right)\sinh\left(\gamma V_k\right). \qquad (3)$$

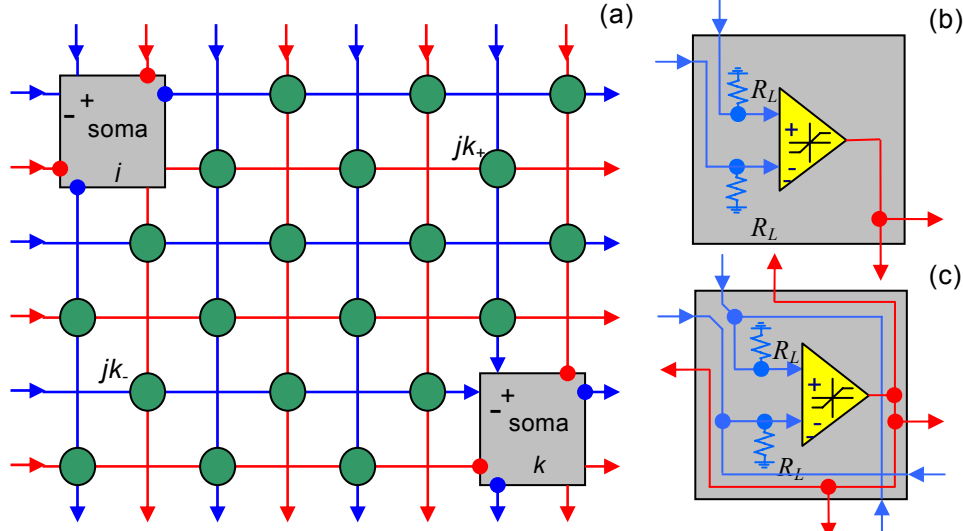

Fig. 3. (a) Generic structure of the simplest, (feedforward, non-Hebbian) CrossNet. Red lines show "axonic", and blue lines "dendritic" nanowires. Gray squares are interfaces between nanowires and CMOS-based somas (b, c). Signs show the dendrite input polarities. Green circles denote molecular latching switches forming elementary synapses. Bold red and blue points are open-circuit terminations of the nanowires, that do not allow somas to interact in bypass of synapses

In the simplest cases (e.g., quasi-Hopfield networks with finite connectivity), the tri-level synaptic weights of the generic CrossNets are quite satisfactory, leading to just a very modest (~30%) network capacity loss. However, some applications (in particular, pattern classification) may require a larger number of weight quantization levels $L$ (e.g., $L \approx 30$ for a 1% fidelity [9]). This may be achieved by using compact square arrays (e.g., 4×4) of latching switches (Fig. 4).

Various species of CrossNets [9] differ also by the way the somatic cells are distributed around the synaptic field. Figure 5 shows feedforward versions of two CrossNet types most explored so far: the so-called FlossBar and InBar. The former network is more natural for the implementation of multilayered perceptrons (MLP), while the latter system is preferable for recurrent network implementations and also allows a simpler CMOS design of somatic cells.

The most important advantage of CrossNets over the hardware neural networks suggested earlier is that these networks allow to achieve enormous density combined with large cell connectivity $M \gg 1$ in quasi-2D electronic circuits.

## 4   CrossNet training

CrossNet training faces several hardware-imposed challenges:

(i) The synaptic weight contribution provided by the elementary latching switch is binary, so that for most applications the multi-switch synapses (Fig. 4) are necessary.

(ii) The only way to adjust any particular synaptic weight is to turn ON or OFF the corresponding latching switch(es). This is only possible to do by applying certain voltage $V = V_j - V_k$ between the two corresponding nanowires. At this procedure, other nanodevices attached to the same wires should not be disturbed.

(iii) As stated above, synapse state switching is a statistical progress, so that the degree of its "fuzziness" should be carefully controlled.

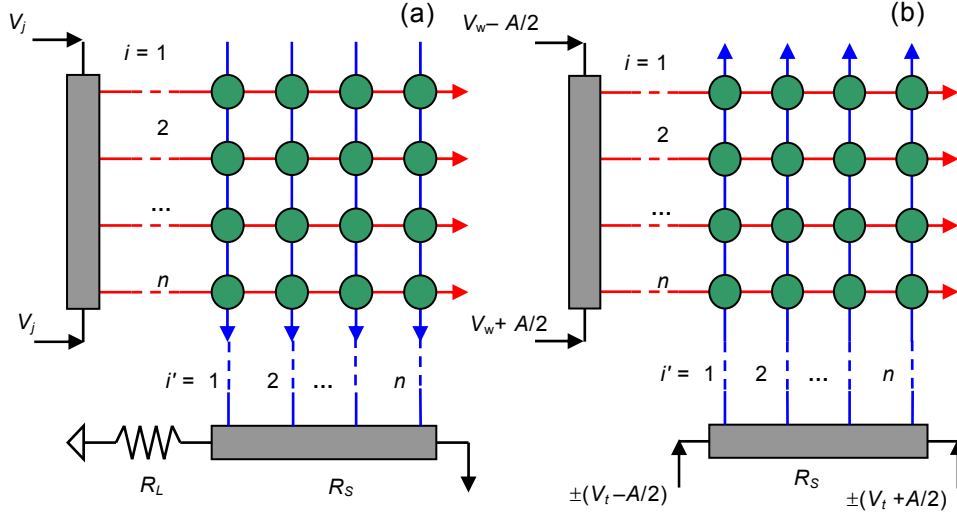

Fig. 4. Composite synapse for providing $L = 2n^2 + 1$ discrete levels of the weight in (a) operation and (b) weight adjustment modes. The dark-gray rectangles are resistive metallic strips at soma/nanowire interfaces

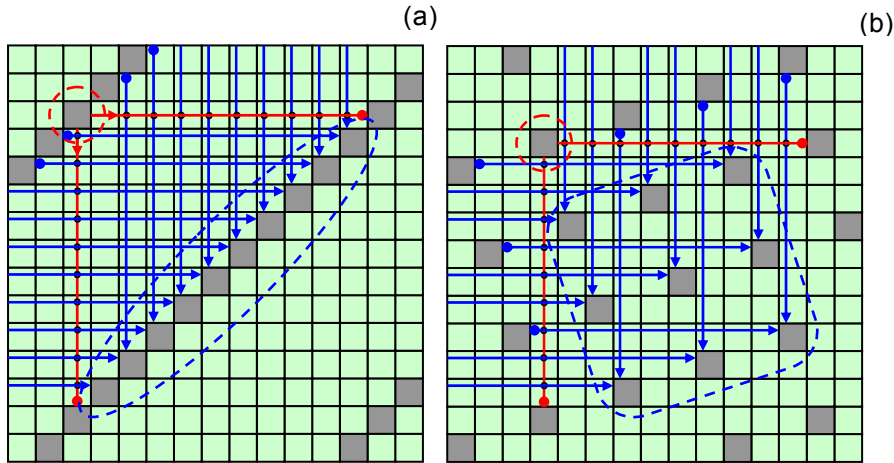

Fig. 5. Two main CrossNet species: (a) FlossBar and (b) InBar, in the generic (feedforward, non-Hebbian, ternary-weight) case for the connectivity parameter $M = 9$. Only the nanowires and nanodevices coupling one cell (indicated with red dashed lines) to $M$ post-synaptic cells (blue dashed lines) are shown; actually all the cells are similarly coupled

We have shown that these challenges may be met using (at least) the following training methods [9]:

(i) *Synaptic weight import.* This procedure is started with training of a homomorphic "precursor" artificial neural network with continuous synaptic weighs $w_{jk}$, implemented in software, using one of established methods (e.g., error backpropagation). Then the synaptic weights $w_{jk}$ are transferred to the CrossNet, with some "clipping" (rounding) due to the binary nature of elementary synaptic weights. To accomplish the transfer, pairs of somatic cells are sequentially selected via CMOS-level wiring. Using the flexibility of CMOS circuitry, these cells are reconfigured to apply external voltages $\pm V_W$ to the axonic and dendritic nanowires leading to a particular synapse, while all other nanowires are grounded. The voltage level $V_W$ is selected so that it does not switch the synapses attached to only one of the selected nanowires, while voltage $2V_W$ applied to the synapse at the crosspoint of the selected wires is sufficient for its reliable switching. (In the composite synapses with quasi-continuous weights (Fig. 4), only a part of the corresponding switches is turned ON or OFF.)

(ii) *Error backpropagation.* The synaptic weight import procedure is straightforward when $w_{jk}$ may be simply calculated, e.g., for the Hopfield-type networks. However, for very large CrossNets used, e.g., as pattern classifiers the precursor network training may take an impractically long time. In this case the direct training of a CrossNet may become necessary. We have developed two methods of such training, both based on "Hebbian" synapses consisting of 4 elementary synapses (latching switches) whose average weight dynamics obeys Eq. (3). This quasi-Hebbian rule may be used to implement the backpropagation algorithm either using a periodic time-multiplexing [9] or in a continuous fashion, using the simultaneous propagation of signals and errors along the same dual-rail channels.

As a result, presently we may state that CrossNets may be taught to perform virtually all major functions demonstrated earlier with the usual neural networks, including the corrupted pattern restoration in the recurrent quasi-Hopfield mode and pattern classification in the feedforward MLP mode [11].

# 5  CrossNet performance estimates

The significance of this result may be only appreciated in the context of unparalleled physical parameters of CMOL CrossNets. The only fundamental limitation on the half-pitch $F_{nano}$ (Fig. 1) comes from quantum-mechanical tunneling between nanowires. If the wires are separated by vacuum, the corresponding specific leakage conductance becomes uncomfortably large ($\sim 10^{-12}$ $\Omega^{-1}m^{-1}$) only at $F_{nano} = 1.5$ nm; however, since realistic insulation materials ($SiO_2$, etc.) provide somewhat lower tunnel barriers, let us use a more conservative value $F_{nano} = 3$ nm. Note that this value corresponds to $10^{12}$ elementary synapses per $cm^2$, so that for $4M = 10^4$ and $n = 4$ the areal density of neural cells is close to $2 \times 10^7$ $cm^{-2}$. Both numbers are higher than those for the human cerebral cortex, despite the fact that the quasi-2D CMOL circuits have to compete with quasi-3D cerebral cortex.

With the typical specific capacitance of $3 \times 10^{-10}$ F/m = 0.3 aF/nm, this gives nanowire capacitance $C_0 \approx 1$ aF per working elementary synapse, because the corresponding segment has length $4F_{nano}$. The CrossNet operation speed is determined mostly by the time constant $\tau_0$ of dendrite nanowire capacitance recharging through resistances of open nanodevices. Since both the relevant conductance and capacitance increase similarly with $M$ and $n$, $\tau_0 \approx R_0 C_0$.

The possibilities of reduction of $R_0$, and hence $\tau_0$, are limited mostly by acceptable power dissipation per unit area, that is close to $V_s^2/(2F_{nano})^2 R_0$. For room-temperature operation, the voltage scale $V_0 \approx V_t$ should be of the order of at least 30 $k_B T/e \approx 1$ V to avoid thermally-induced errors [9]. With our number for $F_{nano}$, and a relatively high but acceptable power consumption of 100 W/$cm^2$, we get $R_0 \approx 10^{10}\Omega$ (which is a very realistic

value for single-molecule single-electron devices like one shown in Fig. 3). With this number, $\tau_0$ is as small as ~10 ns. This means that the CrossNet speed may be approximately six orders of magnitude (!) higher than that of the biological neural networks. Even scaling $R_0$ up by a factor of 100 to bring power consumption to a more comfortable level of 1 W/cm$^2$, would still leave us at least a four-orders-of-magnitude speed advantage.

## 6  Discussion: Possible applications

These estimates make us believe that that CMOL CrossNet chips may revolutionize the neuromorphic network applications. Let us start with the example of relatively small (1-cm$^2$-scale) chips used for recognition of a face in a crowd [11]. The most difficult feature of such recognition is the search for face location, i.e. optimal placement of a face on the image relative to the panel providing input for the processing network. The enormous density and speed of CMOL hardware gives a possibility to time-and-space multiplex this task (Fig. 6). In this approach, the full image (say, formed by CMOS photodetectors on the same chip) is divided into $P$ rectangular panels of $h \times w$ pixels, corresponding to the expected size and approximate shape of a single face. A CMOS-implemented communication channel passes input data from each panel to the corresponding CMOL neural network, providing its shift in time, say using the TV scanning pattern (red line in Fig. 6). The standard methods of image classification require the network to have just a few hidden layers, so that the time interval $\Delta t$ necessary for each mapping position may be so short that the total pattern recognition time $T = hw\Delta t$ may be acceptable even for online face recognition.

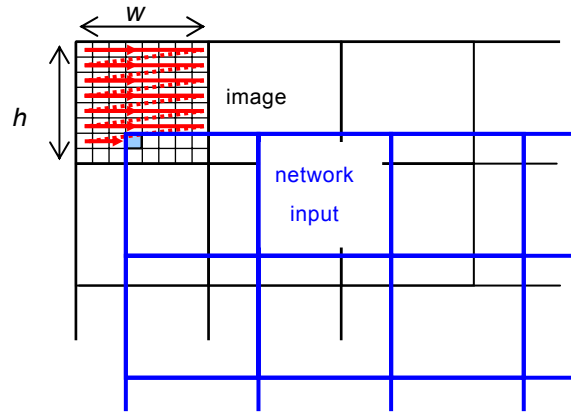

Fig. 6. Scan mapping of the input image on CMOL CrossNet inputs. Red lines show the possible time sequence of image pixels sent to a certain input of the network processing image from the upper-left panel of the pattern

Indeed, let us consider a 4-Megapixel image partitioned into 4K 32×32-pixel panels ($h = w = 32$). This panel will require an MLP net with several (say, four) layers with 1K cells each in order to compare the panel image with ~10$^3$ stored faces. With the feasible 4-nm nanowire half-pitch, and 65-level synapses (sufficient for better than 99% fidelity [9]), each interlayer crossbar would require chip area about (4K×64 nm)$^2$ = 64×64 μm$^2$, fitting 4×4K of them on a ~0.6 cm$^2$ chip. (The CMOS somatic-layer and communication-system overheads are negligible.) With the acceptable power consumption of the order of 10 W/cm$^2$, the input-to-output signal propagation in such a network will take only about 50 ns, so that $\Delta t$ may be of the order of 100 ns and the total time $T = hw\Delta t$ of processing one frame of the order of 100 microseconds, much shorter than the typical TV frame time of ~10 milliseconds. The remaining

two-orders-of-magnitude time gap may be used, for example, for double-checking the results via stopping the scan mapping (Fig. 6) at the most promising position. (For this, a simple feedback from the recognition output to the mapping communication system is necessary.)

It is instructive to compare the estimated CMOL chip speed with that of the implementation of a similar parallel network ensemble on a CMOS signal processor (say, also combined on the same chip with an array of CMOS photodetectors). Even assuming an extremely high performance of 30 billion additions/multiplications per second, we would need $\sim 4 \times 4K \times 1K \times (4K)^2/(30 \times 10^9) \approx 10^4$ seconds $\sim 3$ hours per frame, evidently incompatible with the online image stream processing.

Let us finish with a brief (and much more speculative) discussion of possible long-term prospects of CMOL CrossNets. Eventually, large-scale ($\sim 30 \times 30$ cm$^2$) CMOL circuits may become available. According to the estimates given in the previous section, the integration scale of such a system (in terms of both neural cells and synapses) will be comparable with that of the human cerebral cortex. Equipped with a set of broadband sensor/actuator interfaces, such (necessarily, hierarchical) system may be capable, after a period of initial supervised training, of further self-training in the process of interaction with environment, with the speed several orders of magnitude higher than that of its biological prototypes. Needless to say, the successful development of such self-developing systems would have a major impact not only on all information technologies, but also on the society as a whole.

## Acknowledgments

This work has been supported in part by the AFOSR, MARCO (via FENA Center), and NSF. Valuable contributions made by Simon Fölling, Özgür Türel and Ibrahim Muckra, as well as useful discussions with P. Adams, J. Barhen, D. Hammerstrom, V. Protopopescu, T. Sejnowski, and D. Strukov are gratefully acknowledged.

## References

[1] Frank, D. J. *et al.* (2001) Device scaling limits of Si MOSFETs and their application dependencies. *Proc. IEEE* **89**(3): 259-288.

[2] Likharev, K. K. (2003) Electronics below 10 nm, in J. Greer et al. (eds.), *Nano and Giga Challenges in Microelectronics*, pp. 27-68. Amsterdam: Elsevier.

[3] Likharev, K. K. and Strukov, D. B. (2005) CMOL: Devices, circuits, and architectures, in G. Cuniberti *et al.* (eds.), *Introducing Molecular Electronics*, Ch. 16. Springer, Berlin.

[4] Fölling, S., Türel, Ö. & Likharev, K. K. (2001) Single-electron latching switches as nanoscale synapses, in *Proc. of the 2001 Int. Joint Conf. on Neural Networks*, pp. 216-221. Mount Royal, NJ: Int. Neural Network Society.

[5] Wang, W. *et al.* (2003) Mechanism of electron conduction in self-assembled alkanethiol monolayer devices. *Phys. Rev. B* **68**(3): 035416 1-8.

[6] Stan M. *et al.* (2003) Molecular electronics: From devices and interconnect to circuits and architecture, *Proc. IEEE* **91**(11): 1940-1957.

[7] Strukov, D. B. & Likharev, K. K. (2005) Prospects for terabit-scale nanoelectronic memories. *Nanotechnology* **16**(1): 137-148.

[8] Strukov, D. B. & Likharev, K. K. (2005) CMOL FPGA: A reconfigurable architecture for hybrid digital circuits with two-terminal nanodevices. *Nanotechnology* **16**(6): 888-900.

[9] Türel, Ö. *et al.* (2004) Neuromorphic architectures for nanoelectronic circuits", *Int. J. of Circuit Theory and Appl.* **32**(5): 277-302.

[10] See, e.g., Hertz J. *et al.* (1991) *Introduction to the Theory of Neural Computation*. Cambridge, MA: Perseus.

[11] Lee, J. H. & Likharev, K. K. (2005) CrossNets as pattern classifiers. *Lecture Notes in Computer Sciences* **3575:** 434-441.
